# Bayesian Models of Inductive Generalization

**Neville E. Sanjana & Joshua B. Tenenbaum**
Department of Brain and Cognitive Sciences
Massachusetts Institute of Technology
Cambridge, MA 02139
{nsanjana, jbt}@mit.edu

## Abstract

We argue that human inductive generalization is best explained in a Bayesian framework, rather than by traditional models based on similarity computations. We go beyond previous work on Bayesian concept learning by introducing an unsupervised method for constructing flexible hypothesis spaces, and we propose a version of the Bayesian Occam's razor that trades off priors and likelihoods to prevent under- or over-generalization in these flexible spaces. We analyze two published data sets on inductive reasoning as well as the results of a new behavioral study that we have carried out.

## 1 Introduction

The problem of inductive reasoning — in particular, how we can generalize after seeing only one or a few specific examples of a novel concept — has troubled philosophers, psychologists, and computer scientists since the early days of their disciplines. Computational approaches to inductive generalization range from simple heuristics based on similarity matching to complex statistical models [5]. Here we consider where human inference falls on this spectrum. Based on two classic data sets from the literature and one more comprehensive data set that we have collected, we will argue for models based on a rational Bayesian learning framework [10]. We also confront an issue that has often been side-stepped in previous models of concept learning: the origin of the learner's hypothesis space. We present a simple, unsupervised clustering method for creating hypotheses spaces that, when applied to human similarity judgments and embedded in our Bayesian framework, consistently outperforms the best alternative models of inductive reasoning based on similarity-matching heuristics.

We focus on two related inductive generalization tasks introduced in [6], which involve reasoning about the properties of animals. The first task is to judge the strength of a generalization from one or more specific kinds of mammals to a different kind of mammal: given that animals of kind $A$ and $B$ have property $Z$, how likely is it that an animal of kind $C$ also has property $Z$? For example, $A$ might be *chimp*, $B$ might be *squirrel*, and $C$ might be *horse*. $Z$ is always a blank predicate, such as "is susceptible to the disease blicketitis", about which nothing is known outside of the given examples. Working with blank predicates ensures that people's inductions are driven by their deep knowledge about the general features of animals rather than the details they might or might not know about any

one particular property. Stimuli are typically presented in the form of an argument from premises (examples) to conclusion (the generalization test item), as in

> Chimps are susceptible to the disease blicketitis.
> Squirrels are susceptible to the disease blicketitis.
> ___________________________________________
> Horses are susceptible to the disease blicketitis.

and subjects are asked to judge the strength of the argument — the likelihood that the conclusion (below the line) is true given that the premises (above the line) are true. The second task is the same except for the form of the conclusion. Instead of asking how likely the property is to hold for another kind of mammal, e.g., *horses*, we ask how likely it is to hold for *all mammals*. We refer to these two kinds of induction tasks as the *specific* and *general* tasks, respectively.

Osherson et al. [6] present data from two experiments using these tasks. One data set contains human judgments for the relative strengths of 36 specific inferences, each with a different pair of mammals given as examples (premises) but the same test species, *horses*. The other set contains judgments of argument strength for 45 general inferences, each with a different triplet of mammals given as examples and the same test category, *all mammals*. Osherson et al. also published subjects' judgments of similarity for all 45 pairs of the 10 mammals used in their generalization experiments, which they (and we) use to build models of generalization.

## 2  Previous approaches

There have been several attempts to model the data in [6]: the similarity-coverage model [6], a feature-based model [8], and a Bayesian model [3]. The two factors that determine the strength of an inductive generalization in Osherson et al.'s model [6] are (i) *similarity* of the animals in the premise(s) to those in the conclusion, and (ii) *coverage*, defined as the similarity of the animals in the premise(s) to the larger taxonomic category of mammals, including all specific animal types in this domain. To see the importance of the coverage factor, compare the following two inductive generalizations. The chance that horses can get a disease given that we know chimps and squirrels can get that disease seems higher than if we know only that chimps and gorillas can get the disease. Yet simple similarity favors the latter generalization: horses are judged to be more similar to gorillas than to chimps, and much more similar to either primate species than to squirrels. Coverage, however, intuitively favors the first generalization: the set {*chimp*, *squirrel*} "covers" the set of all mammals much better than does the set {*chimp*, *gorilla*}, and to the extent that a set of examples supports generalization to all mammals, it should also support generalization to horses, a particular type of mammal.

Similarity and coverage factors are mixed linearly to predict the strength of a generalization. Mathematically, the prediction is given by $\alpha R(X, Y) + (1 - \alpha) R(X, \text{all mammals})$, where $X$ is the set of examples (premises), $Y$ is the test set (conclusion), $\alpha$ is a free parameter, and $R(A, B)$ is a setwise similarity metric defined to be the sum of each $B$ element's maximal similarity to the $A$ elements: $R(A, B) = \sum_j \max_i sim(A_i, B_j)$. For the specific arguments, the test set $Y$ has just one element, $y = horse$, so $R(X, Y)$ is just the maximum similarity of horses to the example animal types in $X$. For the general arguments, $Y = all\ mammals$, which is approximated by the set of all mammal types used in the experiment (see Figure 1). Osherson et al. [6] also consider a sum-similarity model, which replaces the maximum with a sum: $R(A, B) = \sum_j \sum_i sim(A_i, B_j)$. Summed similarity has more traditionally been used to model human concept learning, and also has a rational interpretation in terms of nonparametric density estimation, but Osherson et al. favor the

max-similarity model based on its match to their intuitions for these particular tasks. We examine both models in our experiments.

Sloman [8] developed a feature-based model that encodes the shared features between the premise set and the conclusion set as weights in a neural network. Despite some psychological plausibility, this model consistently fit the two data sets significantly worse than the max-similarity model. Heit [3] outlines a Bayesian framework that provides qualitative explanations of various inductive reasoning phenomena from [6]. His model does not constrain the learner's hypothesis space, nor does it embody a generative model of the data, so its predictions depend strictly on well-chosen prior probabilities. Without a general method for setting these prior probabilities, it does not make quantitative predictions that can be compared here.

## 3  A Bayesian model

Tenenbaum & colleagues have previously introduced a Bayesian framework for learning concepts from examples, and applied it to learning number concepts [10], word meanings [11], as well as other domains. Formally, for the specific inference task, we observe $n$ positive examples $X = \{x^{(1)}, \ldots, x^{(n)}\}$ of the concept $C$ and want to compute the probability that a particular test stimulus $y$ belongs to the concept $C$ given the observed examples $X$: $p(y \in C | X)$. These generalization probabilities $p(y \in C | X)$ are computed by averaging the predictions of a set of hypotheses weighted by their posterior probabilities:

$$p(y \in C | X) = \sum_{h \in \mathcal{H}} p(y \in C | h) p(h | X) = \sum_{h : y \in h} p(h | X). \tag{1}$$

Hypotheses $h$ pick out subsets of stimuli — candidate extensions of the concept $C$ — and $p(y \in C | h)$ is just 1 or 0 depending on whether the test stimulus $y$ falls under the subset $h$. In the general inference task, we are interested in computing the probability that a whole test category $Y$ falls under the concept $C$:

$$p(Y \subset C | X) = \sum_{h : Y \subset h} p(h | X). \tag{2}$$

A crucial component in modeling both tasks is the structure of the learner's hypothesis space $\mathcal{H}$.

### 3.1  Hypothesis space

Elements of the hypothesis space $\mathcal{H}$ represent natural subsets of the objects in the domain — subsets likely to be the extension of some novel property or concept. Our goal in building up $\mathcal{H}$ is to capture as many hypotheses as possible that people might employ in concept learning, using a procedure that is ideally automatic and unsupervised. One natural way to begin is to identify hypotheses with the clusters returned by a clustering algorithm [11][7].

Here, hierarchical clustering seems particularly appropriate, as people across cultures appear to organize their concepts of biological species in a hierarchical taxonomic structure [1]. We applied four standard agglomerative clustering algorithms [2] (single-link, complete-link, average-link, and centroid) to subjects' similarity judgments for all pairs of 10 animals given in [6]. All four algorithms produced the same output (Figure 1), suggesting a robust cluster structure. We define the base set of clusters $\mathcal{B}$ to consist of all 19 clusters in this tree. The most straightforward way to define a hypothesis space for Bayesian concept learning is to take $\mathcal{H}_1 = \mathcal{B}$; each hypothesis consists of one base cluster. We refer to $\mathcal{H}_1$ as the "taxonomic hypothesis space".

It is clear that $\mathcal{H}_1$ alone is not sufficient. The chance that horses can get a disease given that we know cows and squirrels can get that disease seems much higher than if we know only

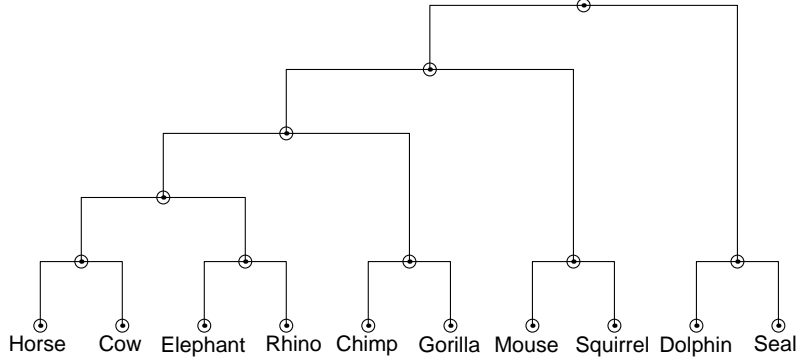

Figure 1: Hierarchical clustering of mammals based on similarity judgments in [6]. Each node in the tree corresponds to one hypothesis in the taxonomic hypothesis space $\mathcal{H}_1$.

that chimps and squirrels can get the disease, yet the taxonomic hypotheses consistent with the example sets {*cow*, *squirrel*} and {*chimp*, *squirrel*} are the same. Bayesian generalization with a purely taxonomic hypothesis space essentially depends only on the *least* similar example (here, *squirrel*), ignoring more fine-grained similarity structure, such as that one example in the set {*cow*, *squirrel*} is very similar to the target *horse* even if the other is not. This sense of fine-grained similarity has a clear objective basis in biology, because a single property can apply to more than one taxonomic cluster, either by chance or through convergent evolution. If the disease in question could afflict two distinct clusters of animals, one exemplified by cows and the other by squirrels, then it is much more likely also to afflict horses (since they share most taxonomic clusters with cows) than if the disease afflicted two distinct clusters exemplified by chimps and squirrels. Thus we consider richer hypothesis subspaces $\mathcal{H}_2$, consisting of all pairs of taxonomic clusters (i.e., all unions of two clusters from Figure 1, except those already included in $\mathcal{H}_1$), and $\mathcal{H}_3$, consisting of all triples of taxonomic clusters (except those included in lower layers). We stop with $\mathcal{H}_3$ because we have no behavioral data beyond three examples. Our total hypothesis space is then the union of these three layers, $\mathcal{H} = \mathcal{H}_1 \cup \mathcal{H}_2 \cup \mathcal{H}_3$.

The notion that the hypothesis space of candidate concepts might correspond to the power set of the base clusters, rather than just single clusters, is broadly applicable beyond the domain of biological properties. If the base system of clusters is sufficiently fine-grained, this framework can parameterize any logically possible concept. It is analogous to other general-purpose representations for concepts, such as disjunctive normal form (DNF) in PAC-Learning, or class-conditional mixture models in density-based classification [5].

### 3.2 The Bayesian Occam's razor: balancing priors and likelihoods

Given this hypothesis space, Bayesian generalization then requires assigning a prior $p(h)$ and likelihood $p(X|h)$ for each hypothesis $h \in \mathcal{H}$. Let $|\mathcal{B}|$ be the number of base clusters, and $h$ be a hypothesis in the $k$th layer of the hypothesis space $\mathcal{H}_k$, corresponding to a union of $k$ base clusters. A simple but reasonable prior assigns to $h$ a sequence of $|\mathcal{B}|$ i. i. d. Bernoulli variables with $k$ successes and parameter $\lambda$, with probability

$$p(h) \propto \lambda^k(1-\lambda)^{|\mathcal{B}|-k} \propto \left(\frac{\lambda}{1-\lambda}\right)^k . \tag{3}$$

Intuitively, this choice of prior is like assuming a generative model for hypotheses in which each base cluster has some small independent probability $\lambda$ of expressing the concept $C$;

the correspondence is not exact because each hypothesis may be expressed as the union of base clusters in multiple ways, and we consider only the minimal union in defining $p(h)$. For $\lambda < 1/2$, $p(h)$ instantiates a preference for simpler hypotheses — that is, hypotheses consisting of fewer disjoint clusters (smaller $k$). More complex hypotheses receive exponentially lower probability under $p(h)$, and the penalty for complexity increases as $\lambda$ becomes smaller. This prior can be applied with any set of base clusters, not just those which are taxonomically structured. We are currently exploring a more sophisticated domain-specific prior for taxonomic clusters defined by a stochastic mutation process over the branches of the tree.

Following [10], the likelihood $p(X|h)$ is calculated by assuming that the examples $X$ are a random sample (with replacement) of instances from the concept to be learned. Let $n = |X|$, the number of examples, and let the size $|h|$ of each hypothesis $h$ be simply the number of animal types it contains. Then $p(X|h)$ follows the *size principle*,

$$p(X|h) = \begin{cases} \left[\frac{1}{|h|}\right]^n & \text{if } h \text{ includes all examples in } X \\ 0 & \text{if } h \text{ does not include all examples in } X \end{cases} \tag{4}$$

assigning greater likelihood to smaller hypotheses, by a factor that increases exponentially as the number of consistent examples observed increases.

Note the tension between priors and likelihoods here, which implements a form of the Bayesian Occam's razor. The prior favors hypotheses consisting of few clusters, while the likelihood favors hypotheses consisting of small clusters. These factors will typically trade off against each other. For any set of examples, we can always cover them under a single cluster if we make the cluster large enough, and we can always cover them with a hypothesis of minimal size (i.e., including no other animals beyond the examples) if we use only singleton clusters and let the number of clusters equal the number of examples. The posterior probability $p(h|X)$, proportional to the product of these terms, thus seeks an optimal tradeoff between over- and under-generalization.

## 4  Model results

We consider three data sets. Data sets 1 and 2 come from the specific and general tasks in [6], described in Section 1. Both tasks drew their stimuli from the same set of 10 mammals shown in Figure 1. Each data set (including the set of similarity judgments used to construct the models) came from a different group of subjects. Our models of the probability of generalization for specific and general arguments are given by Equations 1 and 2, respectively, letting $X$ be the example set that varies from trial to trial and $y$ or $Y$ (respectively) be the fixed test category, *horses* or *all mammals*. Osherson at al.'s subjects did not provide an explicit judgment of generalization for each example set, but only a relative ranking of the strengths of all arguments in the general or specific sets. Hence we also converted all models' predictions to ranks for each data set, to enable the most natural comparisons between model and data.

Figure 3 shows the (rank) predictions of three models, Bayesian, max-similarity and sum-similarity, versus human subjects' (rank) confirmation judgments on the general (row 1) and specific (row 2) induction tasks from [6]. Each model had one free parameter ($\lambda$ in the Bayesian model, $\alpha$ in the similarity models), which was tuned to the single value that maximized rank-order correlation between model and data jointly over both data sets.

The best correlations achieved by the Bayesian model in both the general and specific tasks were greater than those achieved by either the max-similarity or sum-similarity models. The sum-similarity model is far worse than the other two — it is actually negatively correlated with the data on the general task — while max-similarity consistently scores slightly worse than the Bayesian model.

### 4.1 A new experiment: Varying example set composition

In order to provide a more comprehensive test of the models, we conducted a variant of the specific experiment using the same 10 animal types and the same constant test category, *horses*, but with example sets of different sizes and similarity structures. In both data sets 1 and 2, the number of examples was constant across all trials; we expected that varying the number of examples would cause difficulty for the max-similarity model because it is not explicitly sensitive to this factor. For this purpose, we included five three-premise arguments, each with three examples of the same animal species (e.g., {*chimp, chimp, chimp*}), and five one-premise arguments with the same five animals (e.g., {*chimp*}). We also included three-premise arguments where all examples were drawn from a low-level cluster of species in Figure 1 (e.g., {*chimp, gorilla, chimp*}). Because of the increasing preference for smaller hypotheses as more examples are observed, Bayes will in general make very different predictions in these three cases, but max-similarity will not. This manipulation also allowed us to distinguish the predictions of our Bayesian model from alternative Bayesian formulations [5][3] that do not include the size principle, and thus do not predict differences between generalization from one example and generalization from three examples of the same kind.

We also changed the judgment task and cover story slightly, to match more closely the natural problem of inductive learning from randomly sampled examples. Subjects were told that they were training to be veterinarians, by observing examples of particular animals that had been diagnosed with novel diseases. They were required to judge the probability that *horses* could get the same disease given the examples observed. This cover story made it clear to subjects that when multiple examples of the same animal type were presented, these instances referred to distinct individual animals. Figure 3 (row 3) shows the model's predicted generalization probabilities along with the data from our experiment: mean ratings of generalization from 24 subjects on 28 example sets, using either $n = 1, 2,$ or $3$ examples and the same test species (*horses*) across all arguments. Again we show predictions for the best values of the free parameters $\lambda$ and $\alpha$. All three models fit best at different parameter values than in data sets 1 and 2, perhaps due to the task differences or the greater range of stimuli here.

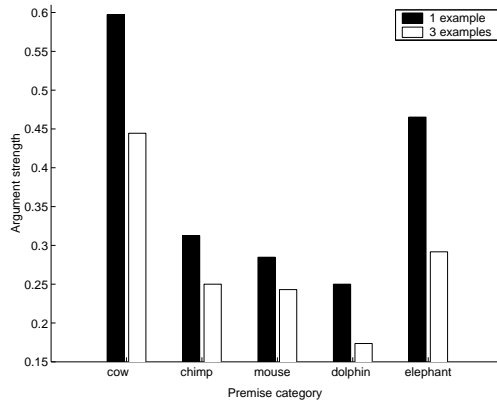

Figure 2: Human generalization to the conclusion category *horse* when given one or three examples of a single premise type.

Again, the max-similarity model comes close to the performance of the Bayesian model, but it is inconsistent with several qualitative trends in the data. Most notably, we found a difference between generalization from one example and generalization from three examples of the same kind, in the direction predicted by our Bayesian model. Generalization to the test category of *horses* was greater from singleton examples (e.g., {*chimp*}) than from three examples of the same kind (e.g., {*chimp, chimp, chimp*}), as shown in Figure 2. This effect was relatively small but it was observed for all five animal types tested and it was

statistically significant ($p < 0.05$) in a $2 \times 5$ (number of examples $\times$ animal type) ANOVA. The max-similarity model, however, predicts no effect here, as do Bayesian accounts that do not include the size principle [5][3].

It is also of interest to ask whether these models are sufficiently robust as to make reasonable predictions across all three experiments using a single parameter setting, or to make good predictions on held-out data when their free parameter is tuned on the remaining data. On these criteria, our Bayesian model maintains its advantage over max-similarity. At the single value of $\lambda = 0.06$, Bayes achieves correlations of $\rho = 0.91, 0.94,$ and $0.97$ on the three data sets, respectively, compared to $\rho = 0.87, 0.90,$ and $0.90$ for max-similarity at its single best parameter value ($\alpha = 0.40$). Using Monte Carlo cross validation [9] to estimate $\lambda$ (1000 runs for each data set, 80%-20% training-test splits), Bayes obtains average test-set correlations of $\rho = 0.90, 0.92$ and $0.92$ on the three data sets, respectively, compared to $\rho = 0.83, 0.83$ and $0.84$ for max-similarity using the same method to tune $\alpha$.

## 5 Conclusion

Our Bayesian model offers a moderate but consistent quantitative advantage over the best similarity-based models of generalization, and also predicts qualitative effects of varying sample size that contradict alternative approaches. More importantly, our Bayesian approach has a principled rational foundation, and we have introduced a framework for unsupervised construction of hypothesis spaces that could be applied in many other domains. In contrast, the similarity-based approach requires arbitrary assumptions about the form of the similarity measure: it must include both "similarity" and "coverage" terms, and it must be based on max-similarity rather than sum-similarity. These choices have no a priori justification and run counter to how similarity models have been applied in other domains, leading us to conclude that rational statistical principles offer the best hope for explaining how people can generalize so well from so little data. Still, the consistently good performance of the max-similarity model raises an important question for future study: whether a relatively small number of simple heuristics might provide the algorithmic machinery implementing approximate rational inference in the brain.

We would also like to understand how people's subjective hypothesis spaces have their origin in the objective structure of their environment. Two plausible sources for the taxonomic hypothesis space used here can both be ruled out. The actual biological taxonomy for these 10 animals, based on their evolutionary history, looks quite different from the subjective taxonomy used here. Substituting the true taxonomic clusters from biology for the base clusters of our model's hypothesis space leads to dramatically worse predictions of people's generalization behavior. Taxonomies constructed from linguistic co-occurrences, by applying the same agglomerative clustering algorithms to similarity scores output from the LSA algorithm [4], also lead to much worse predictions. Perhaps the most likely possibility has not yet been tested. It may well be that by clustering on simple perceptual features (e.g., size, shape, hairiness, speed, etc.), weighted appropriately, we can reproduce the taxonomy constructed here from people's similarity judgments. However, that only seems to push the problem back, to the question of what defines the appropriate features and feature weights. We do not offer a solution here, but merely point to this question as perhaps the most salient open problem in trying to understand the computational basis of human inductive inference.

**Acknowledgments**

Tom Griffiths provided valuable help with statistical analysis. Supported by grants from NTT Communication Science Laboratories and MERL and an HHMI fellowship to NES.

# References

[1] S. Atran. Classifying nature across cultures. In *An Invitation to Cognitive Science*, volume 3. MIT Press, 1995.

[2] R. Duda, P. Hart, and D. Stork. *Pattern Classification*. Wiley, New York, NY, 2001.

[3] E. Heit. A Bayesian analysis of some forms of induction. In *Rational Models of Cognition*. Oxford University Press, 1998.

[4] T. Landauer and S. Dumais. A solution to Plato's problem: The Latent Semantic Analysis theory of the acquisition, induction, and representation of knowledge. *Psychological Review*, 104:211–240, 1997.

[5] T. Mitchell. *Machine Learning*. McGraw-Hill, Boston, MA, 1997.

[6] D. Osherson, E. Smith, O. Wilkie, A. López, and E. Shafir. Category-based induction. *Psychological Review*, 97(2):185–200, 1990.

[7] N. Sanjana and J. Tenenbaum. Capturing property-based similarity in human concept learning. In *Sixth International Conference on Cognitive and Neural Systems*, 2002.

[8] S. Sloman. Feature-based induction. *Cognitive Psychology*, 25:231–280, 1993.

[9] P. Smyth. Clustering using Monte Carlo cross-validation. In *Second International Conference on Knowledge Discovery and Data Mining*, 1996.

[10] J. Tenenbaum. Rules and similarity in concept learning. In S. Solla, T. Keen, and K.-R. Müller, editors, *Advances in Neural Information Processing Systems 12*, pages 59–65. MIT Press, 2000.

[11] J. Tenenbaum and F. Xu. Word learning as Bayesian inference. In *Proceedings of the 22nd Annual Conference of the Cognitive Science Society*, 2000.

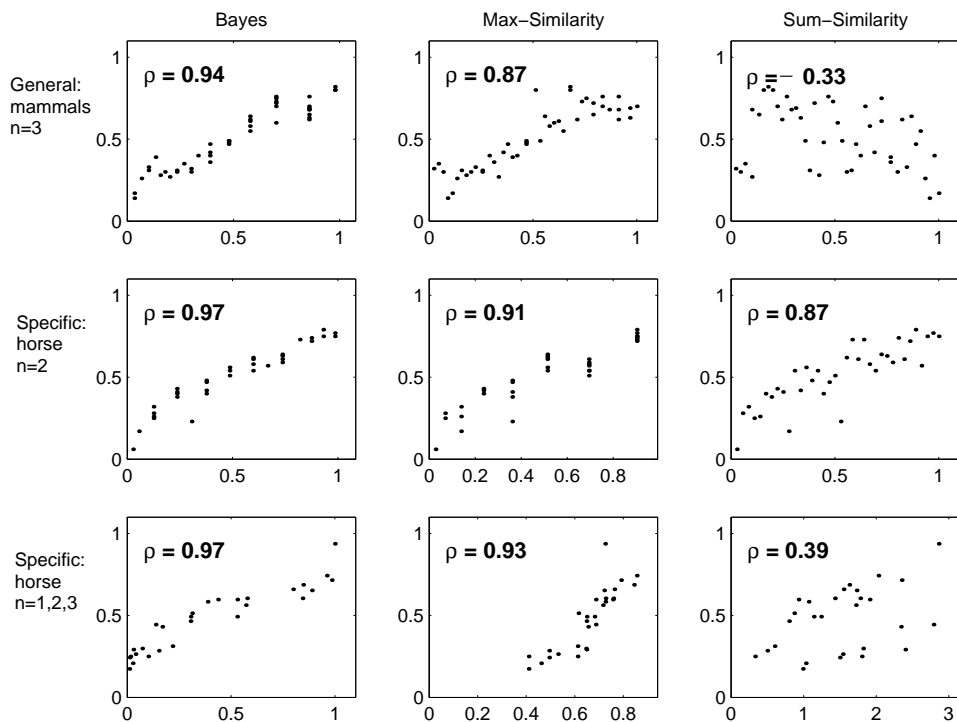

Figure 3: Model predictions ($x$-axis) plotted against human confirmation scores ($y$-axis). Each column shows the results for a particular model. Each row is a different inductive generalization experiment, where $n$ indicates the number of examples (premises) in the stimuli.